# Discriminability-Based Transfer between Neural Networks

**L. Y. Pratt**
Department of Mathematical and Computer Sciences
Colorado School of Mines
Golden, CO 80401
lpratt@mines.colorado.edu

## Abstract

Previously, we have introduced the idea of neural network *transfer*, where learning on a *target* problem is sped up by using the weights obtained from a network trained for a related *source* task. Here, we present a new algorithm, called *Discriminability-Based Transfer* (DBT), which uses an information measure to estimate the utility of hyperplanes defined by source weights in the target network, and rescales transferred weight magnitudes accordingly. Several experiments demonstrate that target networks initialized via DBT learn significantly faster than networks initialized randomly.

## 1 INTRODUCTION

Neural networks are usually trained from scratch, relying only on the training data for guidance. However, as more and more networks are trained for various tasks, it becomes reasonable to seek out methods that avoid "reinventing the wheel", and instead are able to build on previously trained networks' results. For example, consider a speech recognition network that was only trained on American English speakers. However, for a new application, speakers might have a British accent. Since these tasks are sub-distributions of the same larger distribution (English speakers), they may be related in a way that can be exploited to speed up learning on the British network, compared to when weights are randomly initialized.

We have previously introduced the question of how trained neural networks can be

"recycled' in this way [Pratt *et al.*, 1991]; we've called this the *transfer* problem. The idea of transfer has strong roots in psychology (as discussed in [Sharkey and Sharkey, 1992]), and is a standard paradigm in neurobiology, where synapses almost always come "pre-wired".

There are many ways to formulate the transfer problem. Retaining performance on the source task may or may not be important. When it is, the problem has been called *sequential learning*, and has been explored by several authors (cf. [McCloskey and Cohen, 1989]). Our paradigm assumes that source task performance is not important, though when the source task training data is a subset of the target training data, our method may be viewed as addressing sequential learning as well. Transfer knowledge can also be inserted into several different *entry points* in a back-propagation network (see [Pratt, 1993a]). We focus on changing a network's initial weights; other studies change other aspects, such as the objective function (cf. [Thrun and Mitchell, 1993, Naik *et al.*, 1992]).

Transfer methods may or may not use back-propagation for target task training. Our formulation does, because this allows it to degrade, in the worst case of no source task relevance, to back-propagation training on the target task with randomly initialized weights. An alternative approach is described by [Agarwal *et al.*, 1992].

Several studies have explored *literal* transfer in back-propagation networks, where the final weights from training on a source task are used as the initial conditions for target training (cf. [Martin, 1988]). However, these studies have shown that often networks will demonstrate worse performance after literal transfer than if they had been randomly initialized.

This paper describes the Discriminability-Based Transfer (DBT) algorithm, which overcomes problems with literal transfer. DBT achieves the same asymptotic accuracy as randomly initialized networks, and requires substantially fewer training updates. It is also superior to literal transfer, and to just using the source network on the target task.

## 2  ANALYSIS OF LITERAL TRANSFER

As mentioned above, several studies have shown that networks initialized via literal transfer give worse asymptotic performance than randomly initialized networks. To understand why, consider the situation when only a subset of the source network input-to-hidden (IH) layer hyperplanes are relevant to the target problem, as illustrated in Figure 1. We've observed that some hyperplanes initialized by source network training don't shift out of their initial positions, despite the fact that they don't help to separate the target training data. The weights defining such hyperplanes often have high magnitudes [Dewan and Sontag, 1990]. Figure 2 (a) shows a simulation of such a situation, where a hyperplane that has a high magnitude, as if it came from a source network, causes learning to be slowed down.[1]

Analysis of the back-propagation weight update equations reveals that high source weight magnitudes retard back-propagation learning on the target task because this

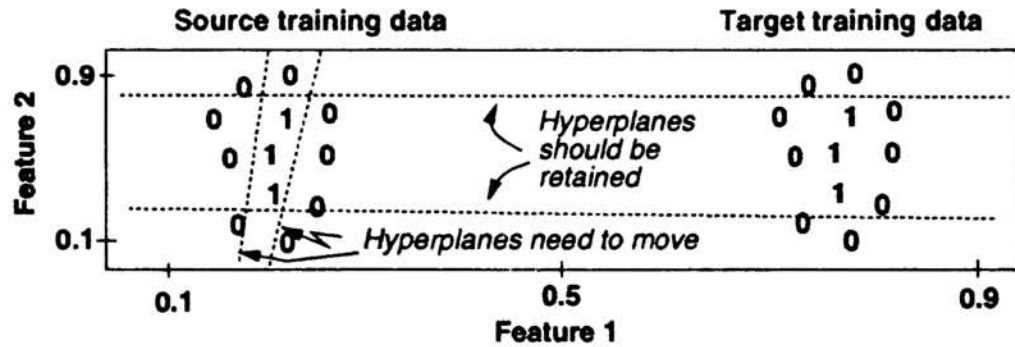

Figure 1: Problem Illustrating the need for DBT. The source and target tasks are identical, except that the target task has been shifted along one axis, as represented by the training data shown. Because of this shift, two of the source hyperplanes are helpful in separating class-0 from class-1 data in the target task, and two are not.

equation is not scaled relative to weight magnitudes. Also, the weight update equation contains the factor $y(1 - y)$ (where $y$ is a unit's activation), which is small for large weights. Considering this analysis, it might at first appear that a simple solution to the problem with literal transfer is to uniformly lower all weight magnitudes. However, we have also observed that hyperplanes in separating positions will move unless they are given high weight magnitudes. To address both of these problems, we must rescale hyperplanes so that useful ones are defined by high-magnitude weights and less useful hyperplanes receive low magnitudes. To implement such a method, we need a metric for evaluating hyperplane utility.

## 3  EVALUATING CLASSIFIER COMPONENTS

We borrow the *IM* metric for evaluating hyperplanes from decision tree induction [Quinlan, 1983]. Given a set of training data and a hyperplane that crosses through it, the IM function returns a value between 0 and 1, indicating the amount that the hyperplane helps to separate the data into different classes.

The formula for IM, for a decision surface in a multi-class problem, is: $\text{IM} = \frac{1}{N} ( \sum \sum x_{ij} \log x_{ij} - \sum x_{i.} \log x_{i.} - \sum x_{.j} \log x_{.j} + N \log N )$ [Mingers, 1989]. Here, $N$ is the number of patterns, $i$ is either 0 or 1, depending on the side of a hyperplane on which a pattern falls, $j$ indexes over all classes, $x_{ij}$ is the count of class $j$ patterns on side $i$ of the hyperplane, $x_{i.}$ is the count of all patterns on side $i$, and $x_{.j}$ is the total number of patterns in class $j$.

## 4  THE DBT ALGORITHM

The DBT algorithm is shown in Figure 3. It inputs the target training data and weights from the source network, along with two parameters $C$ and $S$ (see below). DBT outputs a modified set of weights, for initializing training on the target task. Figure 2 (b) shows how the problem of Figure 2 (a) was repaired via DBT.

DBT modifies the weights defining each source hyperplane to be proportional to the

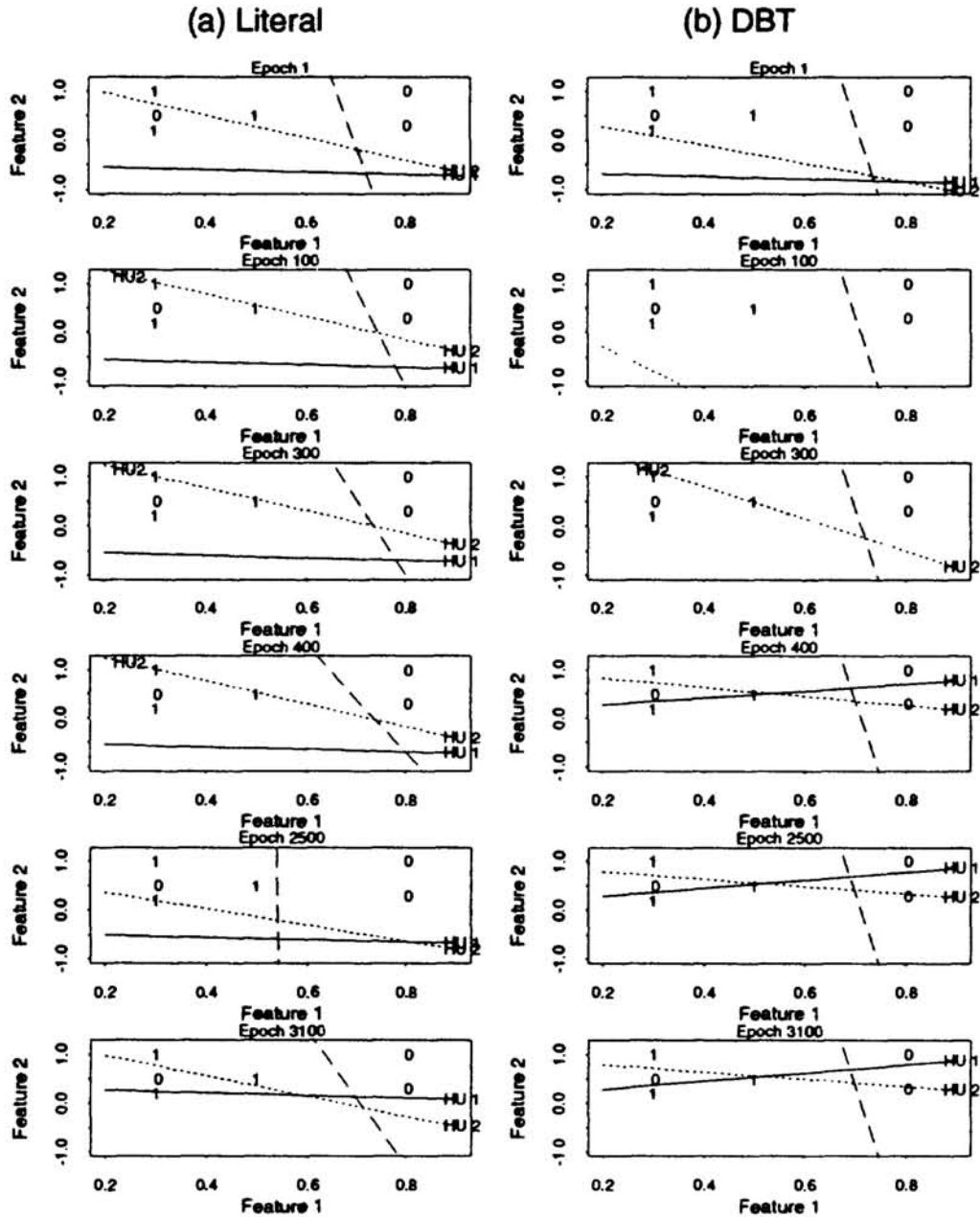

Figure 2: Hyperplane Movement speed in Literal Transfer. Compared to DBT. Each image in this figure shows the hyperplanes implemented by IH weights at a different epoch of training. Hidden unit 1's hyperplane is a solid line; HU2's is a dotted line, and HU3's hyperplane is shown as a dashed line. In (a) note how HU1 seems fixed in place. Its high magnitude causes learning to be slow (taking about 3100 epochs to converge). In (b) note how DBT has given HU1 a small magnitude, allowing it to be flexible, so that the training data is separated by epoch 390. A randomly initialized network on this problem takes about 600 epochs.

**Input:**
> *Source network weights*
> *Target training data*
> *Parameters: $C$ (cutoff factor), $S$ (scaleup factor)*

**Output:**
> *Initial weights for target network, assuming same topology as source network*

**Method:**
> *For each source network hidden unit $i$*
>> *Compare the hyperplane defined by incoming weights to $i$ to the target training data, calculating $IM_{ti}$ ($\epsilon$ $[0,1]$)*
>
> *Rescale $IM_{ti}$ values so that largest has value $S$. Put result in $s_i$.*
> *For $IM_{ti}$'s that are less than $C$*
>> *If highest magnitude ratio between weights defining hyperplane $i$ is $> 100.0$, reset weights for that hyperplane randomly*
>> *Else uniformly scale down hyperplane to have low-valued weights (maximum magnitude of 0.5), but to be in the same position.*
>
> *For each remaining IH hidden unit $i$*
>> *For each weight $w^t_{ji}$ defining hyperplane $i$ in target network*
>>> *Let $w^t_{ji} = $ source weight $w^s_{ji} \times s_i$*
>
> *Set hidden-to-output target network weights randomly in $[-0.5, 0.5]$*

Figure 3: The Discriminability-Based Transfer (DBT) Algorithm.

IM value, according to an input parameter, $S$. DBT is based on the idea that the best initial magnitude $M_t$ for a target hyperplane is $M_t = S \times M_s \times IM_t$, where $S$ ("scaleup") is a constant of proportionality, $M_s$ is the magnitude of a source network hyperplane, and $IM_t$ is the discriminability of the source hyperplane on the target training data. We assume that this simple relationship holds over some range of $IM_t$ values. A second parameter, $C$, determines a cut-off in this relationship – source hyperplanes with $IM_t < C$ receive very low magnitudes, so that the hyperplanes are effectively equivalent to those in a randomly initialized network. The use of the $C$ parameter was motivated by empirical experiments that indicated that the multiplicative scaling via $S$ was not adequate.

To determine $S$ and $C$ for a particular source and target task, we ran DBT several times for a small number of epochs with different $S$ and $C$ values. We chose the $S$ and $C$ values that yielded the best average TSS (total sum of squared errors) after a few epochs. We used local hill climbing in average TSS space to decide how to move in $S$, $C$ space.

DBT randomizes the weights in the network's hidden-to-output (HO) layer. See [Sharkey and Sharkey, 1992] for an extension to this work showing that literal transfer of HO weights might also be effective.

## 5   EMPIRICAL RESULTS

DBT was evaluated on seven tasks: female-to-male speaker transfer on a 10-vowel recognition task (PB), a 3-class subset of the PB task (PB123), transfer from all females to a single male in the PB task (Onemale), transfer for a heart disease diagnosis problem from Hungarian to Swiss patients (Heart-HS), transfer for the same task from patients in California to Swiss patients (Heart-VAS), transfer from a subset of DNA pattern recognition examples to a superset (DNA), and transfer

from a subset of chess endgame problems to a superset (Chess). Note that the DNA and chess tasks effectively address the sequential learning problem; as long as the source data is a subset of the target data, the target network can build on the previous results.

DBT was compared to randomly initialized networks on the target task. We measured generalization performance in both conditions by using 10-way cross-validation on 10 different initial conditions for each target task, resulting in 100 different runs for each of the two conditions, and for each of the seven tasks. Our empirical methodology controlled carefully for initial conditions, hidden unit count, back-propagation parameters $\eta$ (learning rate) and $\alpha$ (momentum), and DBT parameters $S$ and $C$.

## 5.1  SCENARIOS FOR EVALUATION

There are at least two different practical situations in which we may want to speed up learning. First, we may have a limited amount of computer time, all of which will be used because we have no way of detecting when a network's performance has reached some criterion. In this case, if our speed-up method (i.e. DBT) is significantly superior to a baseline for a large proportion of epochs during training, then the probability that we'll have to stop during that period of significant superiority is high If we do stop at an epoch when our method is significantly better, then this justifies it over the baseline, because the resulting network has better performance.

A second situation is when we have some way of detecting when performance is "good enough" for an application. In contrast to the above situation, here a DBT network may be run for a shorter time than a baseline network, because it reaches this criterion faster. In this case, the number of epochs of DBT significant superiority is less important than the speed with which it achieves the criterion.

## 5.2  RESULTS

To evaluate networks according to the first scenario, we tested for statistical significance at the 99.0% level between the 100 DBT and the 100 randomly initialized networks at each training epoch. We found (1) that asymptotic DBT performance scores were the same as for random networks and (2), that DBT was superior for much of the training period. Figure 4 (a) shows the number of weight updates for which a significant difference was found for the seven tasks.

For the second scenario, we also found (3) that DBT networks required many fewer epochs to reach a criterion performance score. For this test, we found the last significantly different epoch between the two methods. Then we measured the number of epochs required to reach 98%, 95%, and 66% of that level. The number of weight updates required for DBT and randomly initialized networks to reach the 98% criterion are shown in Figure 4 (b). Note that the $y$ axis is logarithmic, so, for example, over 30 million weight updates were saved by using DBT instead of random initialization in the PB123 problem. Results for the 95% and 66% criteria also showed DBT to be at least as fast as random initialization for every task.

Using the same tests described for DBT above, we also tested literal networks on the seven transfer tasks. We found that, unlike DBT, literal networks reached sig-

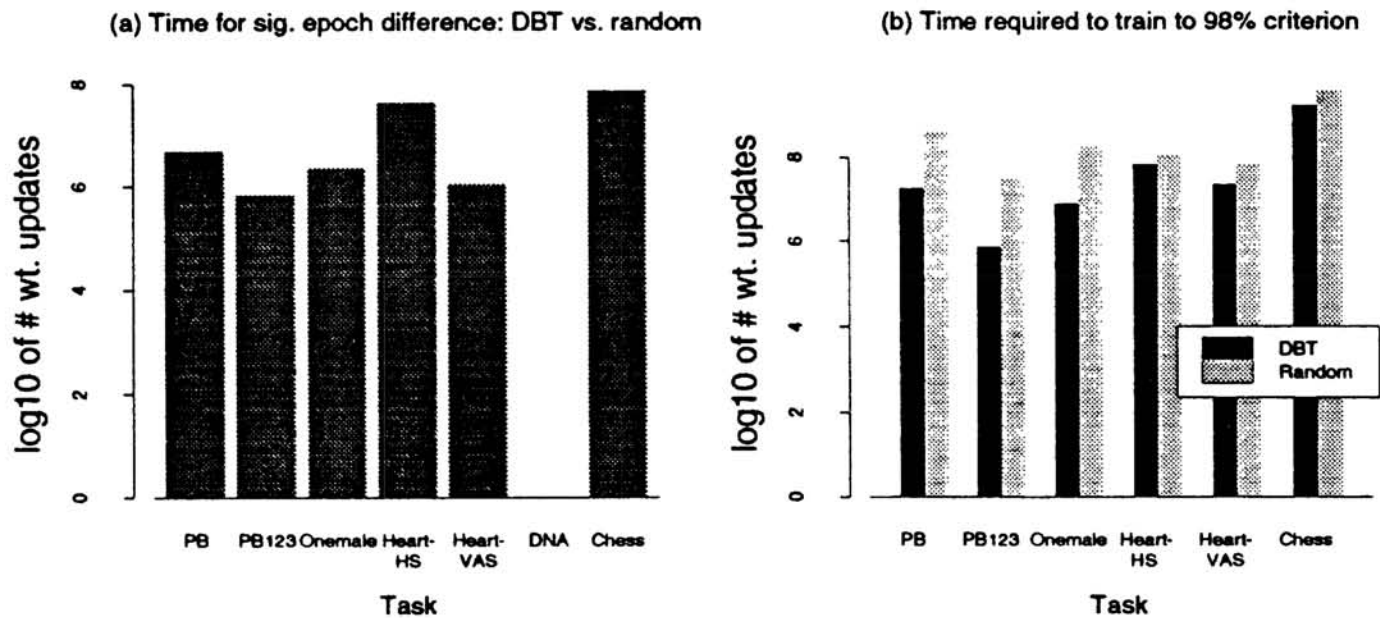

Figure 4: Summary of DBT Empirical Results.

nificantly worse asymptotic performance scores than randomly initialized networks. Literal networks also learned slower for some tasks. These results justify the use of the more complicated DBT method over literal transfer.

We also evaluated the source networks directly on the target tasks, without any back-propagation training on the target training data. Scores were significantly and substantially worse than random networks. This result indicates that the transfer scenarios we chose for evaluation were nontrivial.

# 6    CONCLUSION

We have described the DBT algorithm for transfer between neural networks.[2] DBT demonstrated substantial and significant learning speed improvement over randomly initialized networks in 6 out of 7 tasks studied (and the same learning speed in the other task). DBT never displayed worse asymptotic performance than a randomly initialized network. We have also shown that DBT is superior to literal transfer, and to simply using the source network on the target task.

## Acknowledgements

The author is indebted to John Smith. Gale Martin, and Anshu Agarwal for their valuable comments on this paper, and to Jack Mostow and Haym Hirsh for their contribution to this research program.

## Footnotes

[1] Neural network visualization will be explored more thoroughly in an upcoming paper. An X-based animator is available from the author via anonymous ftp. Type "archie ha".

[2]See [Pratt, 1993b] for more details.

# References

[Agarwal *et al.*, 1992] A. Agarwal, R. J. Mammone, and D. K. Naik. An on-line training algorithm to overcome catastrophic forgetting. In *Intelligence Engineering Systems through Artificial Neural Networks*. volume 2, pages 239–244. The American Society of Mechanical Engineers, ASME Press, 1992.

[Dewan and Sontag, 1990] Hasanat M. Dewan and Eduardo Sontag. Using extrapolation to speed up the backpropagation algorithm. In *Proceedings of the International Joint Conference on Neural Networks, Washington, DC*, volume 1, pages 613–616. IEEE Publications, Inc., January 1990.

[Martin, 1988] Gale Martin. The effects of old learning on new in Hopfield and Backpropagation nets. Technical Report ACA-HI-019. Microelectronics and Computer Technology Corporation (MCC), 1988.

[McCloskey and Cohen, 1989] Michael McCloskey and Neal J. Cohen. Catastrophic interference in connectionist networks: the sequential learning problem. *The psychology of learning and motivation*, 24, 1989.

[Mingers, 1989] John Mingers. An empirical comparison of selection measures for decision- tree induction. *Machine Learning*, 3(4):319–342, 1989.

[Naik *et al.*, 1992] D. K. Naik, R. J. Mammone. and A. Agarwal. Meta-neural network approach to learning by learning. In *Intelligence Engineering Systems through Artificial Neural Networks*, volume 2. pages 245–252. The American Society of Mechanical Engineers, ASME Press. 1992.

[Pratt *et al.*, 1991] Lorien Y. Pratt, Jack Mostow. and Candace A. Kamm. Direct transfer of learned information among neural networks. In *Proceedings of the Ninth National Conference on Artificial Intelligence (AAAI-91)*, pages 584–589, Anaheim, CA, 1991.

[Pratt, 1993a] Lorien Y. Pratt. Experiments in the transfer of knowledge between neural networks. In S. Hanson, G. Drastal, and R. Rivest, editors, *Computational Learning Theory and Natural Learning Systems. Constraints and Prospects*, chapter 4.1. MIT Press, 1993. To appear.

[Pratt, 1993b] Lorien Y. Pratt. Non-literal transfer of information among inductive learners. In R.J.Mammone and Y. Y. Zeevi. editors. *Neural Networks: Theory and Applications II*. Academic Press, 1993. To appear.

[Quinlan, 1983] J. R. Quinlan. Learning efficient classification procedures and their application to chess end games. In *Machine Learning*, pages 463–482. Palo Alto, CA: Tioga Publishing Company. 1983.

[Sharkey and Sharkey, 1992] Noel E. Sharkey and Amanda J. C. Sharkey. Adaptive generalisation and the transfer of knowledge. 1992. Working paper, Center for Connection Science, University of Exeter, 1992.

[Thrun and Mitchell, 1993] Sebastian B. Thrun and Tom M. Mitchell. Integrating inductive neural network learning and explanation-based learning. In C.L. Giles, S. J. Hanson, and J. D. Cowan, editors. *Advances in Neural Information Processing Systems 5*. Morgan Kaufmann Publishers. San Mateo, CA, 1993.